# Matrix Completion from Power-Law Distributed Samples

**Raghu Meka, Prateek Jain, and Inderjit S. Dhillon**
Department of Computer Sciences
University of Texas at Austin
Austin, TX 78712
{raghu,pjain,inderjit}@cs.utexas.edu

## Abstract

The low-rank matrix completion problem is a fundamental problem with many important applications. Recently, [4],[13] and [5] obtained the first non-trivial theoretical results for the problem assuming that the observed entries are sampled uniformly at random. Unfortunately, most real-world datasets do not satisfy this assumption, but instead exhibit power-law distributed samples. In this paper, we propose a graph theoretic approach to matrix completion that solves the problem for more realistic sampling models. Our method is simpler to analyze than previous methods with the analysis reducing to computing the threshold for *complete cascades* in random graphs, a problem of independent interest. By analyzing the graph theoretic problem, we show that our method achieves exact recovery when the observed entries are sampled from the Chung-Lu-Vu model, which can generate power-law distributed graphs. We also hypothesize that our algorithm solves the matrix completion problem from an optimal number of entries for the popular preferential attachment model and provide strong empirical evidence for the claim. Furthermore, our method is easy to implement and is substantially faster than existing methods. We demonstrate the effectiveness of our method on random instances where the low-rank matrix is sampled according to the prevalent random graph models for complex networks and present promising preliminary results on the Netflix challenge dataset.

## 1   Introduction

Completing a matrix from a few given entries is a fundamental problem with many applications in machine learning, statistics, and compressed sensing. Since completion of arbitrary matrices is not a well-posed problem, it is often assumed that the underlying matrix comes from a restricted class. Here we address the matrix completion problem under the natural assumption that the underlying matrix is low-rank.

Formally, for an unknown matrix $M \in \mathbb{R}^{m \times n}$ of rank at most $k$, given $\Omega \subseteq [m] \times [n]$, $\mathcal{P}_\Omega(M)$[1] and $k$, the low-rank matrix completion problem is to find a matrix $X \in \mathbb{R}^{m \times n}$ such that

$$rank(X) \leq k \quad and \quad \mathcal{P}_\Omega(X) = \mathcal{P}_\Omega(M). \tag{1.1}$$

Recently Candes and Recht [4], Keshavan et.al [13], Candes and Tao [5] obtained the first non-trivial guarantees for the above problem under a few additional assumptions on the matrix $M$ and the set of known entries $\Omega$. At a high level, the assumptions made in the above papers can be stated as follows.

   A1   $M$ is *incoherent*, in the sense that the singular vectors of $M$ are not correlated with the standard basis vectors.

A2 The observed entries are sampled uniformly at random.

In this work we address some of the issues with assumption [A2]. For $\Omega \subseteq [m] \times [n]$, let the *sampling graph* $G_\Omega = (U, V, \Omega)$ be the bipartite graph with vertices $U = \{u_1, \ldots, u_m\}$, $V = \{v_1, \ldots, v_n\}$ and edges given by the ordered pairs in $\Omega$ [2]. Then, assumption [A2] can be reformulated as follows:

A3 The sampling graph $G_\Omega$ is an Erdős-Rényi random graph[3].

A prominent feature of Erdős-Rényi graphs is that the degrees of vertices are Poisson distributed and are sharply concentrated about their mean. The techniques of [4, 5], [13], as will be explained later, crucially rely on these properties of Erdős-Rényi graphs. However, for most large real-world graphs such as the World Wide Web ([1]), the degree distribution deviates significantly from the Poisson distribution and has high variance. In particular, most large matrix-completion datasets such as the much publicized Netflix prize dataset and the Yahoo Music dataset exhibit power-law distributed degrees, i.e., the number of vertices of degree $d$ is proportional to $d^{-\beta}$ for a constant $\beta$ (Figure 1).

In this paper, we overcome some of the shortcomings of assumption [A3] above by considering more realistic random graph models for the sampling graph $G_\Omega$. We propose a natural graph theoretic approach for matrix completion (referred to as ICMC for *information cascading matrix completion*) that we prove can handle sampling graphs with power-law distributed degrees. Our approach is motivated by the models for information *cascading* in social networks proposed by Kempe et al. [11, 12]. Moreover, the analysis of ICMC reduces to the problem of finding density *thresholds* for *complete cascades* in random graphs - a problem of independent interest.

By analyzing the threshold for complete cascades in the random graph model of Chung, Lu & Vu [6] (CLV model), we show that ICMC solves the matrix completion problem for sampling graphs drawn from the CLV model. The bounds we obtain for matrix-completion on the CLV model are incomparable to the main results of [4, 5, 13]. The methods of the latter papers do not apply to models such as the CLV model that generate graphs with skewed degrees. On the other hand, for Erdos-Renyi graphs the density requirements for ICMC are stronger than those of the above papers.

We also empirically investigate the threshold for complete cascading in other popular random graph models such as the preferential attachment model [1], the forest-fire model [17] and the affiliation networks model [16]. The empirical estimates we obtain for the threshold for complete cascading in the preferential attachment model strongly suggest that ICMC solves the exact matrix-completion problem from an optimal number of entries for sampling procedures with preferential attachment.

Our experiments demonstrate that for sampling graphs drawn from more realistic models such as the preferential attachment, forest-fire and affiliation network models, ICMC outperforms - both in accuracy and time - the methods of [4, 5, 3, 13] by an order of magnitude.

In summary, our main contributions are:

- We formulate the sampling process in matrix completion as generating random graphs ($G_\Omega$) and demonstrate that the sampling assumption [A3] does not hold for real-world datasets.
- We propose a novel graph theoretic approach to matrix completion (ICMC) that extensively uses the link structure of the sampling graph. We emphasize that previously none of the methods exploited the structure of the sampling graph.
- We prove that our method solves the matrix completion problem exactly for sampling graphs generated from the CLV model which can generate power-law distributed graphs.
- We empirically evaluate our method on more complex random graph models and on the Netflix Challenge dataset demonstrating the effectiveness of our method over those of [4, 5, 3, 13].

## 2 Previous Work and Preliminaries

The Netflix challenge has recently drawn much attention to the low-rank matrix completion problem. Most methods for matrix completion and the more general rank minimization problem with affine constraints are based on either relaxing the non-convex rank function to a convex function or assuming a factorization of the matrix and optimizing the resulting non-convex problem using alternating minimization and its variants [2, 15, 18].

Until recently, most methods for rank minimization subject to affine constraints were heuristic in nature with few known rigorous guarantees. In a recent breakthrough, Recht et.al [20] extend the techniques of compressed sensing to rank minimization with affine constraints. However, the results of Recht et.al do not apply to the case of matrix completion as the constraints in matrix completion do not satisfy the *restricted isoperimetry property* they assume.

Building on the work of Recht et al. [20], Candes and Recht [4] and Candes and Tao [5] showed that minimizing the trace-norm recovers the unknown low-rank matrix exactly under certain conditions. However, these approaches require the observed entries to be sampled uniformly at random and as suggested by our experiments, do not work well when the observed entries are not drawn uniformly.

Independent of [4, 5], Keshavan et al. [13] also obtained similar results for matrix completion using different techniques that generalize the works of Friedman et al. [9], Feige and Ofek [8] on the spectrum of random graphs. However, the results of [13], crucially rely on the *regularity* of Erdős-Rényi graphs and do not extend to sampling graphs with skewed degree distributions even for rank one matrices. This is mainly because the results of Friedman et al. and Feige and Ofek on the *spectral gap* of Erdős-Rényi graphs do not hold for graph models with skewed expected degrees (see [6, 19]).

We also remark that several natural variants of the *trimming* phase of [8] and [13] did not improve the performance in our experiments. A similar observation was made in [19], [10] who address the problem of re-weighting the edges of graphs with skewed degrees in the context of LSA.

## 2.1 Random Graph Models

We focus on four popular models of random graphs all of which can generate graphs with power-law distributed degrees. In contrast to the common descriptions of the models, we need to work with bipartite graphs; however, the models we consider generalize naturally to bipartite graphs. Due to space limitations we only give a (brief) description of the Chung et.al [6], and refer to the original papers for the preferential attachment [1], forest-fire [17] and affiliation networks [16] models.

The CLV model [6] generates graphs with arbitrary *expected degree sequences*, $p_1, \ldots, p_m$, $q_1, \ldots, q_n$ with $p_1 + \ldots + p_m = q_1 + \ldots + q_n = w$. In the model, a bipartite graph $G = (U, V, E)$ with $U = \{u_1, \ldots, u_m\}, V = \{v_1, \ldots, v_n\}$ is generated by independently placing an edge between vertices $u_i, v_j$ with probability $p_i q_j / w$ for all $i \in [m], j \in [n]$. We define the *density* of an instance of CLV model to be the expected average degree $(p_1 + \ldots + p_m)/(mn) = w/mn$.

The CLV model is more general than the standard Erdős-Rényi model with the case $p_i = np, q_i = mp$ corresponding to the standard Erdős-Rényi model with density $p$ for bipartite random graphs. Further, by choosing weights that are power-law distributed, the CLV model can generate graphs with power-law distributed degrees, a prominent feature of real-world graphs.

## 3 Matrix Completion from Information Cascading

We now present our algorithm ICMC. Consider the following standard formulation of the low-rank matrix completion problem: Given $k, \Omega, \mathcal{P}_\Omega(M)$ for a rank $k$ matrix $M$, find $X, Y$ such that

$$\mathcal{P}_\Omega(XY^T) = \mathcal{P}_\Omega(M), \quad X \in \mathbb{R}^{m \times k}, Y \in \mathbb{R}^{n \times k}. \tag{3.1}$$

Note that given $X$ we can find $Y$ and vice versa by solving a linear least squares regression problem. This observation is the basis for the popular *alternate minimization* heuristic and its variants which outperform most methods in practice. However, analyzing the performance of alternate minimization is a notoriously hard problem. Our algorithm can be seen as a more refined version of the alternate minimization heuristic that is more amenable to analysis. We assume that the target matrix $M$ is non-degenerate in the following sense.

**Definition 3.1** *A rank $k$ matrix $Z$ is non-degenerate if there exist $X \in \mathbb{R}^{m \times k}, Y \in \mathbb{R}^{n \times k}$, $Z = XY^T$ such that any $k$ rows of $X$ are linearly independent and any $k$ rows of $Y$ are linearly independent.*

Though reminiscent of the *incoherence* property used by Candes and Recht, Keshavan et al., non-degeneracy appears to be incomparable to the incoherence property used in the above works. Observe that a random low-rank matrix is almost surely non-degenerate.

Our method progressively computes rows of $X$ and $Y$ so that Equation (3.1) is satisfied. Call a vertex $u_i \in U$ as *infected* if the $i$'th row of $X$ has been computed (the term *infected* is used to reflect

that infection spreads by *contact* as in an epidemic). Similarly, call a vertex $v_j \in V$ as infected if the $j$'th row of $Y$ has been computed. Suppose that at an intermediate iteration, vertices $L \subseteq U$ and $R \subseteq V$ are marked as infected. That is, the rows of $X$ with indices in $L$ and rows of $Y$ with indices in $R$ have been computed exactly.

Now, for an uninfected $j \in [n]$, to compute the corresponding row of $Y$, $\boldsymbol{y}_j^T \in \mathbb{R}^k$, we only need $k$ independent linear equations. Thus, if $M$ is non-degenerate, to compute $\boldsymbol{y}_j^T$ we only need $k$ entries of the $j$'th column of $M$ with row indices in $L$. Casting the condition in terms of the sampling graph $G_\Omega$, $\boldsymbol{y}_j^T$ can be computed and vertex $v_j \in V$ be marked as infected if there are at least $k$ edges from $v_j$ to infected vertices in $L$. Analogously, $\boldsymbol{x}_i^T$ can be computed and the vertex $u_i \in U$ be marked as infected if there are at least $k$ edges from $u_i$ to previously infected vertices $R$.

Observe that $M = XY^T = XWW^{-1}Y^T$, for any invertible matrix $W \in \mathbb{R}^{k \times k}$. Thus for non-degenerate $M$, without loss of generality, a set of $k$ rows of $X$ can be fixed to be the $k \times k$ identity matrix $I_k$. This suggests the following *cascading* procedure for infecting vertices in $G_\Omega$ and progressively computing the rows of $X, Y$. Here $L_0 \subseteq U$ with $|L_0| = k$.

---

$\mathsf{ICMC}(G_\Omega, \mathcal{P}_\Omega(M), L_0)$:
  1 Start with initially infected sets $L = L_0 \subseteq U$, $R = \emptyset$. Set the $k \times k$ sub-matrix of $X$ with rows in $L_0$ to be $I_k$.
  2 Repeat until convergence:
      (a) Mark as infected all uninfected vertices in $V$ that have at least $k$ edges to previously infected vertices $L$ and add the newly infected vertices to $R$.
      (b) For each newly infected vertex $v_j \in R$, compute the $j$'th row of $Y$ using the observed entries of $M$ corresponding to edges from $v_j$ to $L$.
      (c) Mark as infected all uninfected vertices in $U$ that have at least $k$ edges to previously infected vertices $R$ and add the newly infected vertices to $L$.
      (d) For each newly infected vertex $u_i \in L$, compute the $i$'th row of $X$ using the observed entries of $M$ corresponding to edges from $u_i$ to $R$
  3 Output $M' = XY^T$.

---

We abstract the cascading procedure from above using the framework of Kempe et al. [11] for information cascades in social networks. Let $G = (W, E)$ be an undirected graph and fix $A \subseteq W$, $k > 0$. Define $\sigma_{G,k}(A, 0) = A$ and for $t > 0$ define $\sigma_{G,k}(A, t + 1)$ inductively by

$$\sigma_{G,k}(A, t + 1) = \sigma_{G,k}(A, t) \cup \{u \in W : u \text{ has at least } k \text{ edges to } \sigma_{G,k}(A, t)\}.$$

**Definition 3.2** *The influence of a set $A \subseteq W$, $\sigma_{G,k}(A)$, is the number of vertices infected by the cascading process upon termination when starting at $A$. That is, $\sigma_{G,k}(A) = |\cup_t \sigma_{G,k}(A, t)|$. We say $A$ is completely cascading of order $k$ if $\sigma_{G,k}(A) = |W|$.*

We remark that using a variant of the standard depth-first search algorithm, the cascading process above can be computed in *linear time* for any set $A$. From the discussion preceding $\mathsf{ICMC}$ it follows that $\mathsf{ICMC}$ recovers $M$ exactly if the cascading process starting at $L_0$ infects all vertices of $G_\Omega$ and we get the following theorem.

**Theorem 3.1** *Let $M$ be a non-degenerate matrix of rank $k$. Then, given $G_\Omega = (U, V, \Omega), \mathcal{P}_\Omega(M)$ and $L_0 \subseteq U$ with $|L_0| = k$, $\mathsf{ICMC}(G_\Omega, \mathcal{P}_\Omega(M), L_0)$ recovers the matrix $M$ exactly if $L_0$ is a completely cascading set of order $k$ in $G_\Omega$.*

Thus, we have reduced the matrix-completion problem to the graph-theoretic problem of finding a completely cascading set (if it exists) in a graph. A more general case of the problem – finding a set of vertices that maximize influence, was studied by Kempe et al. [11] for more general cascading processes. They show the general problem of maximizing influence to be NP-hard and give approximation algorithms for several classes of instances.

However, it appears that for most reasonable random graph models, the highest degree vertices have large influence with high probability. In the following we investigate completely cascading sets in random graphs and show that for CLV graphs, the $k$ highest degree vertices form a completely cascading set with high probability.

# 4 Information Cascading in Random Graphs

We now show that for sufficiently dense CLV graphs and fixed $k$, the $k$ highest degree vertices form a completely cascading set with high probability.

**Theorem 4.1** *For every $\gamma > 0$, there exists a constant $c(\gamma)$ such that the following holds. Consider an instance of the CLV model given by weights $p_1, \ldots, p_m$, $q_1, \ldots, q_n$ with density $p$ and $\min(p_i, q_j) \geq c(\gamma)k \log n/p^k$. Then, for $G = (U, V, E)$ generated from the model, the $k$ highest degree vertices of $U$ form a completely cascading set of order $k$ with probability at least $1 - n^{-\gamma}$.*

**Proof sketch** We will show that the highest weight vertices $L_0 = \{u_1, \ldots, u_k\}$ form a completely cascading set with high probability; the theorem follows from the above statement and the observation that the highest degree vertices of $G$ will almost surely correspond to vertices with large weights in the model; we omit these details for lack of space. Let $w = \sum_i p_i = \sum_j q_j = mnp$ and $m \leq n$.

Fix a vertex $u_i \notin L_0$ and consider an arbitrary vertex $v_j \in V$. Let $P_j^i$ be the indicator variable that is 1 if $(u_i, v_j) \in E$ and $v_j$ is connected to all vertices of $L_0$. Note that vertex $u_i$ will be infected after two rounds by the cascading process starting at $L_0$ if $\sum_j P_j^i \geq k$. Now, $\Pr[P_j^i = 1] = (p_i q_j/w) \prod_{1 \leq l \leq k} (p_l q_j/w)$ and

$$\mathsf{E}[P_1^i + \ldots + P_n^i] = \sum_{j=1}^{n} \frac{p_i q_j}{w} \prod_{l \leq k} \frac{p_l q_j}{w} = \frac{p_i}{w^{k+1}} \cdot \left( \prod_{1 \leq l \leq k} p_l \right) \cdot \sum_{j=1}^{n} q_j^{k+1}. \tag{4.1}$$

Observe that $\sum_i p_i = w \leq nk + p_k(m - k)$. Thus, $p_k \geq (w - nk)/(m - k)$. Now, using the power-mean inequality we get,

$$q_1^{k+1} + q_2^{k+1} + \ldots + q_n^{k+1} \geq n \left( \frac{q_1 + \ldots + q_n}{n} \right)^{k+1} = n \cdot \left( \frac{w}{n} \right)^{k+1}, \tag{4.2}$$

with equality occurring only if $q_j = w/n$ for all $j$. From Equations (4.1), (4.2) we have

$$\mathsf{E}[P_1^i + \ldots + P_n^i] \geq p_i \cdot \left( \frac{w - nk}{m - k} \right)^k \cdot \frac{1}{n^k}$$

$$= p_i \cdot \left( 1 - \frac{nk}{w} \right)^k \cdot \left( 1 - \frac{k}{m} \right)^{-k} \cdot \left( \frac{w}{mn} \right)^k. \tag{4.3}$$

It is easy to check that under our assumptions, $w \geq nk^2$ and $m \geq k^2$. Thus, $(1 - nk/w)^k \geq 1/e$ and $(1 - k/m)^{-k} \geq 1/2e$. From Equation (4.3) and our assumption $p_i \geq c(\gamma)k \log n/p^k$, we get $\mathsf{E}[P_1^i + \ldots + P_n^i] \geq c(\gamma)k \log n/4e^2$.

Now, since the indicator variables $P_1^i, \ldots, P_n^i$ are independent of each other, using the above lower bound for the expectation of their sum and Chernoff bounds we get $\Pr[P_1^i + \ldots + P_n^i \leq k] \leq \exp(-\Omega(c(\gamma) \log n))$. Thus, for a sufficiently large constant $c(\gamma)$, the probability that the vertex $u_i$ is uninfected after two rounds $\Pr[P_1 + \ldots + P_n \leq k] \leq 1/2m^{\gamma+1}$. By taking a union bound over all vertices $u_{k+1}, \ldots, u_m$, the probability that there is an uninfected vertex in the left partition after two steps of cascading starting from $L_0$ is at most $1/2m^{\gamma}$. The theorem now follows by observing that if the left partition is completely infected, for a suitably large constant $c(\gamma)$, all vertices in the right will be infected with probability at least $1 - 1/2m^{\gamma}$ as $q_j \geq c(\gamma)k \log n.\square$

Combining the above with Theorem 3.1 we obtain exact matrix-completion for sampling graphs drawn from the CLV model.

**Theorem 4.2** *Let $M$ be a non-degenerate matrix of rank $k$. Then, for sampling graphs $G_\Omega$ generated from a CLV model satisfying the conditions of Theorem 4.1, ICMC recovers the matrix $M$ exactly with high probability.*

**Remark:** The above results show exact-recovery for CLV graphs with densities up to $n^{-1/k} = o(1)$. As mentioned in the introduction, the above result is incomparable to the main results of [4, 5], [13].

The main bottleneck for the density requirements in the proof of Theorem 4.1 is Equation (4.2) relating $\sum_j q_j^{k+1}$ to $(\sum_j q_j)^{k+1}$, where we used the power-mean inequality. However, when the

expected degrees $q_j$ are skewed, say with a power-law distribution, it should be possible to obtain much better bounds than those of Equation (4.2), hence also improving the density requirements. Thus, in a sense the Erdős-Rényi graphs are the worst-case examples for our analysis.

Our empirical simulations also suggest that completely cascading sets are more likely to exist in random graph models with power-law distributed expected degrees as compared to Erdős-Rényi graphs. Intuitively, this is because of the following reasons.

- In graphs with power-law distributed degrees, the high degree vertices have much higher degrees than the average degree of the graph. So, infecting the highest degree vertices is more likely to infect more vertices in the first step.
- More importantly, as observed in the seminal work of Kleinberg [14] in most real-world graphs there are a small number of vertices (*hubs*) that have much higher connectivity than most vertices. Thus, infecting the *hubs* is likely to infect a large fraction of vertices.

Thus, we expect ICMC to perform better on models that are closer to real-world graphs and have power-law distributed degrees. In particular, as strongly supported by experiments (see Figure 3), we hypothesize that ICMC solves exact matrix completion from an almost optimal number of entries for sampling graphs drawn from the preferential attachment model.

**Conjecture 4.3** *There exists a universal constant $C$ such that for all $k \geq 1$, $k_1, k_2 \geq Ck$ the following holds. For $G = (U, V, E)$ generated from the preferential attachment model with parameters $m, n, k_1, k_2$, the $k$ highest degree vertices of $U$ form a completely cascading set of order $k$ with high probability.*

If true, the above combined with Theorem 3.1 would imply the following.

**Conjecture 4.4** *Let $M$ be a non-degenerate matrix of rank $k$. Then, for sampling graphs $G_\Omega$ generated from a PA model with parameters $k_1, k_2 \geq Ck$, ICMC recovers the matrix $M$ exactly with high probability.*

**Remark:** To solve the matrix completion problem we need to sample at least $(m + n)k$ entries. Thus, the bounds above are optimal up to a constant factor. Moreover, the bounds above are stronger than those obtainable - even information theoretically - for Erdős-Rényi graphs, as for Erdős-Rényi graphs we need to sample $\Omega(n \log n)$ entries even for $k = 1$.

## 5 Experimental Results

We first demonstrate that for many real-world matrix completion datasets, the observed entries are far from being sampled uniformly with the sampling graph having power-law distributed degrees. We then use various random graph models to compare our method against the trace-norm based singular value thresholding algorithm of [3], the spectral matrix completion algorithm (SMC) of [13] and the regularized alternating least squares minimization (ALS) heuristic. Finally, we present empirical results on the Netflix challenge dataset. For comparing with SVT and SMC, we use the code provided by the respective authors; while we use our own implementation for ALS. Below we provide a few implementation details for our algorithm ICMC.

**Implementation Details**

Consider step 2(b) of our algorithm ICMC. Let $L_j$ be the set of vertices in $L$ that have an edge to $v_j$, $L_j^k$ be any size $k$ subset of $L_j$, and let $X(L_j^k, :)$ be the sub-matrix of $X$ containing rows corresponding to vertices in $L_j^k$. If the underlying matrix is indeed low-rank and there is no noise in the observed entries, then for a newly infected vertex $v_j$, the corresponding row of $Y$, $\boldsymbol{y}_j^T$, can be computed by solving the following linear system of equations: $M(L_j^k, j) = X(L_j^k, :)\boldsymbol{y}_j$. To account for noise in measurements, we compute $\boldsymbol{y}_j$ by solving the following regularized least squares problem: $\boldsymbol{y}_j = \text{argmin}_{\boldsymbol{y}} \|M(L_j, j) - X(L_j, :)\boldsymbol{y}\|_2^2 + \lambda \|\boldsymbol{y}\|_2^2$, where $\lambda$ is a regularization parameter. Similarly, we compute $\boldsymbol{x}_i^T$ by solving: $\boldsymbol{x}_i = \text{argmin}_{\boldsymbol{x}} \|M(i, R_i)^T - Y(R_i, :)\boldsymbol{x}\|_2^2 + \lambda \|\boldsymbol{x}\|_2^2$.

Note that if ICMC fails to infect all the vertices, i.e. $L \subsetneq U$ or $R \subsetneq V$, then rows of $X$ and $Y$ will not be computed for vertices in $U \backslash L$ and $V \backslash R$. Let $X = [X_L, X_{\tilde{L}}]$, where $X_L$ is the set of computed rows of $X$ (for vertices in $L$) and $X_{\tilde{L}}$ denotes the remaining rows of $X$. Similarly, let $Y = [Y_R, Y_{\tilde{R}}]$. We estimate $X_{\tilde{L}}$ and $Y_{\tilde{R}}$ using an alternating least squares based heuristic that solves the following:

$$\min_{X_{\tilde{L}}, Y_{\tilde{R}}} \left\| P_\Omega \left( M - \begin{bmatrix} X_L \\ X_{\tilde{L}} \end{bmatrix} [Y_R^T \; Y_{\tilde{R}}^T] \right) \right\|_F^2 + \mu \|X_{\tilde{L}}\|_F^2 + \mu \|Y_{\tilde{R}}\|_F^2,$$

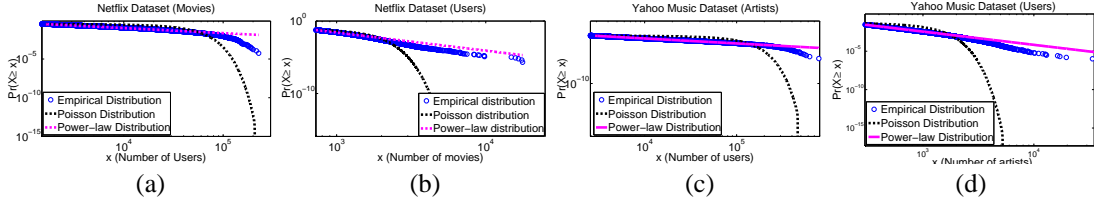

Figure 1: Cumulative degree distribution of (a) movies, (b) users (Netflix dataset) and (c) artists, (d) users (Yahoo Music dataset). Note that degree distributions in all the four cases closely follow power-law distribution and deviate heavily from Poisson-distribution, which is assumed by SVT [3] and SMC [13].

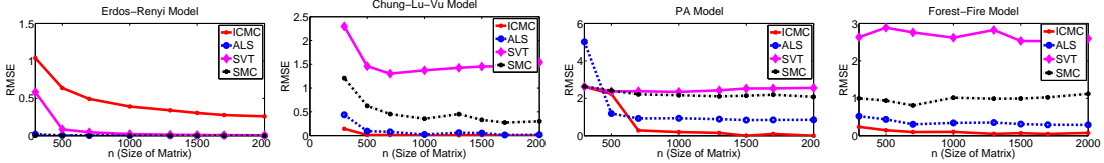

Figure 2: Results on synthetic datasets for fixed sampling density with sampling graph coming from different Graph Models: (a) Erdős-Rényi model, (b) Chung-Lu-Vu model, (c) Preferential attachment model, and (d) Forest-fire model. Note that for the three power-law distribution generating models our method (ICMC) achieves considerably lower RMSE than the existing method.

| (a) Erdős-Rényi Graphs | | | | | (b) Chung-Lu-Vu Graphs | | | | |
|---|---|---|---|---|---|---|---|---|---|
| n/Method | SMC | SVT | ALS | ICMC | n/Method | SMC | SVT | ALS | ICMC |
| 500 | 45.51 | 8.88 | 1.09 | 1.28 | 500 | 35.32 | 14.69 | 1.24 | 0.49 |
| 1000 | 93.85 | 17.07 | 2.39 | 3.30 | 1000 | 144.19 | 17.55 | 2.24 | 2.02 |
| 1500 | 214.65 | 38.81 | 4.85 | 6.28 | 1500 | 443.48 | 30.99 | 3.89 | 3.91 |
| 2000 | 343.76 | 59.88 | 7.20 | 9.89 | 2000 | 836.99 | 46.69 | 5.67 | 5.50 |
| (c) Preferential Attachment Graphs | | | | | (d)Forest-fire Graphs | | | | |
| n/Method | SMC | SVT | ALS | ICMC | n/Method | SMC | SVT | ALS | ICMC |
| 500 | 15.05 | 14.40 | 3.97 | 1.94 | 500 | 22.63 | 5.53 | 0.57 | 0.39 |
| 1000 | 67.96 | 16.49 | 5.06 | 2.01 | 1000 | 85.26 | 11.32 | 1.75 | 1.23 |
| 1500 | 178.35 | 24.48 | 9.83 | 3.65 | 1500 | 186.81 | 21.39 | 3.30 | 2.99 |
| 2000 | 417.54 | 32.06 | 15.07 | 7.46 | 2000 | 350.98 | 27.37 | 4.84 | 5.06 |

Table 1: Time required (in seconds) by various methods on synthetic datasets for fixed sampling density with sampling graph coming from different Graph Models: (a) Erdős-Rényi model, (b) Chung-Lu-Vu model, (c) Preferential attachment model, and (d) Forest-fire model. Note that our method (ICMC) is significantly faster than SVT and SMC, and has similar run-time to that of ALS.

where $\mu \geq 0$ is the regularization parameter.

### Sampling distribution in Netflix and Yahoo Music Datasets

The Netflix challenge dataset contains the incomplete user-movie ratings matrix while the Yahoo Music dataset contains the incomplete user-artist ratings matrix. For both datasets we form the corresponding bipartite sampling graphs and plot the left (users) and right (movies/artists) cumulative degree distributions of the bipartite sampling graphs.

Figure 1 shows the cumulative degree distributions of the bipartite sampling graphs, the best power-law fit computed using the code provided by Clauset et.al [7] and the best Poisson distribution fit. The figure clearly shows that the sampling graphs for the Netflix and Yahoo Music datasets are far from regular as assumed in [4],[5],[13] and have power-law distributed degrees.

### Experiments using Random Graph Models

To compare various methods, we first generate random low-rank matrices $X \in \mathbb{R}^{n \times n}$ for varying $n$, and sample from the generated matrices using Erdős-Rényi, CLV, PA and forest-fire random graph models. We omit the results for the affiliation networks model from this paper due to lack of space; we observed similar trends on the affiliation networks model.

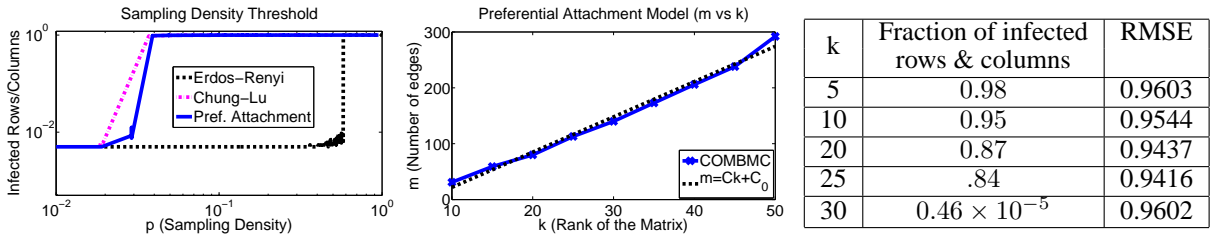

| k | Fraction of infected rows & columns | RMSE |
|---|---|---|
| 5 | 0.98 | 0.9603 |
| 10 | 0.95 | 0.9544 |
| 20 | 0.87 | 0.9437 |
| 25 | .84 | 0.9416 |
| 30 | $0.46 \times 10^{-5}$ | 0.9602 |

Figure 3: **Left**: Fraction of infected nodes as edge density increases. Note the existence of a clear threshold. The threshold is quite small for CLV and PA suggesting good performance of ICMC for these models. **Middle**: Threshold for parameters $k_1, k_2$ (the number of edges per node) in PA as $k$ increases. The threshold varies linearly with $k$ supporting Conjecture 4.3. **Right**: Fraction of infected rows and columns using ICMC for the Netflix challenge dataset.

For each random graph model we compare the relative mean square error (RMSE) on the unknown entries achieved by our method ICMC against several existing methods. We also compare the total time taken by each of the methods. All results represent the average over 20 runs.

Figure 2 compares the RMSE achieved by ICMC to that of SVT, SMC and ALS when rank $k$ is fixed to be 10, sampling density $p = 0.1$, and the sampling graphs are generated from the four random graph models. Note that for the more-realistic CLV, PA, forest-fire three models ICMC outperforms both SVT and SMC significantly and performs noticeably better than ALS. Table 1 compares the computational time taken by each of the methods. The table shows that for all three models, ICMC is faster than SVT and SMC by an order of magnitude and is also competitive to ALS. Note that the performance of our method for Erdos-Renyi graphs (Figure 2 (a)) is poor, with other methods achieving low RMSE. This is expected as the Erdos-Renyi graphs are in a sense the worst-case examples for ICMC as explained in Section 4.

**Threshold for Complete Cascading**

Here we investigate the *threshold* for complete cascading in the random graph models. Besides being interesting on its own, the existence of completely cascading sets is closely tied to the success of ICMC by Theorem 3.1. Figure 3 shows the fraction of vertices infected by the cascading process starting from the $k$ highest degree vertices for graphs generated from the random graph models as the edge density increases.

The left plot of Figure 3 shows the existence of a clear threshold for the density $p$, beyond which the fraction of infected vertices is almost surely one. Note that the threshold is quite small for the CLV, PA and forest-fire models, suggesting good performance of ICMC on these models. As was explained in Section 4, the threshold is bigger for the Erdős-Rényi graph model.

The right plot of Figure 3 shows the threshold value (the minimum value above which the infected fraction is almost surely one) for $k_1, k_2$ as a function of $k$ in the PA model. The plot shows that the threshold is of the form $Ck$ for a universal constant $C$, strongly supporting Conjectures 4.3, 4.4.

**Netflix Challenge Dataset**

Finally, we evaluate our method on the Netflix Challenge dataset which contains an incomplete matrix with about 100 million ratings given by 480,189 users for 17,770 movies. The rightmost table in Figure 3 shows the fraction of rows and columns infected by ICMC on the dataset for several values of the rank parameter $k$. Note that even for a reasonably high rank of 25, ICMC infects a high percentage (84%) of rows and columns. Also, for rank 30 the fraction of infected rows and columns drops to almost zero, suggesting that the sampling density of the matrix is below the sampling threshold for rank 30.

For rank $k = 20$, the RMSE incurred over the probe set (provided by Netflix) is 0.9437 which is comparable to the RMSE=0.9404 achieved by the regularized Alternating Least Squares method. More importantly, the time required by our method is $1.59 \times 10^3$ seconds compared to $6.15 \times 10^4$ seconds required by ALS. We remark that noise (or higher rank of the underlying matrix) can offset our method leading to somewhat inferior results. In such a case, our method can be used for a good initialization of the ALS method and other state-of-the-art collaborative filtering methods to achieve better RMSE.

## Footnotes

[1]Throughout this paper $\mathcal{P}_\Omega : \mathbb{R}^{m \times n} \to \mathbb{R}^{m \times n}$ will denote the projection of a matrix onto the pairs of indices in $\Omega$: $(\mathcal{P}_\Omega(X))_{ij} = X_{ij}$ for $(i,j) \in \Omega$ and $(\mathcal{P}_\Omega(X))_{ij} = 0$ otherwise.

[2]We will often abuse notation and identify edges $(u_i, v_j)$ with ordered pairs $(i, j)$.

[3]We consider the Erdős-Rényi model, where edges $(u_i, v_j) \in E$ independently with probability for $p$ for $(i, j) \in [m] \times [n]$ and $p$ is the density parameter.

# References

[1] Albert-Laszlo Barabasi and Reka Albert. Emergence of scaling in random networks. *Science*, 286:509, 1999.

[2] Matthew Brand. Fast online svd revisions for lightweight recommender systems. In *SDM*, 2003.

[3] Jian-Feng Cai, Emmanuel J. Candes, and Zuowei Shen. A singular value thresholding algorithm for matrix completion, 2008.

[4] Emmanuel J. Candès and Benjamin Recht. Exact matrix completion via convex optimization. *CoRR*, abs/0805.4471, 2008.

[5] Emmanuel J. Candès and Terence Tao. The power of convex relaxation: Near-optimal matrix completion. *CoRR*, abs/0903.1476, 2009.

[6] Fan R. K. Chung, Linyuan Lu, and Van H. Vu. The spectra of random graphs with given expected degrees. *Internet Mathematics*, 1(3), 2003.

[7] A. Clauset, C.R. Shalizi, and M.E.J. Newman. Power-law distributions in empirical data. *SIAM Review*, page to appear, 2009.

[8] Uriel Feige and Eran Ofek. Spectral techniques applied to sparse random graphs. *Random Struct. Algorithms*, 27(2):251–275, 2005.

[9] Joel Friedman, Jeff Kahn, and Endre Szemerédi. On the second eigenvalue in random regular graphs. In *STOC*, pages 587–598, 1989.

[10] Christos Gkantsidis, Milena Mihail, and Ellen W. Zegura. Spectral analysis of internet topologies. In *INFOCOM*, 2003.

[11] David Kempe, Jon M. Kleinberg, and Éva Tardos. Maximizing the spread of influence through a social network. In *KDD*, pages 137–146, 2003.

[12] David Kempe, Jon M. Kleinberg, and Éva Tardos. Influential nodes in a diffusion model for social networks. In *ICALP*, pages 1127–1138, 2005.

[13] Raghunandan H. Keshavan, Sewoong Oh, and Andrea Montanari. Matrix completion from a few entries. *CoRR*, abs/0901.3150, 2009.

[14] Jon M. Kleinberg. Hubs, authorities, and communities. *ACM Comput. Surv.*, 31(4es):5, 1999.

[15] Yehuda Koren. Factorization meets the neighborhood: a multifaceted collaborative filtering model. In *KDD*, pages 426–434, 2008.

[16] Silvio Lattanzi and D. Sivakumar. Affiliation networks. In *STOC*, 2009.

[17] Jure Leskovec, Jon M. Kleinberg, and Christos Faloutsos. Graph evolution: Densification and shrinking diameters. *TKDD*, 1(1), 2007.

[18] Yehuda Koren M. Bell. Scalable collaborative filtering with jointly derived neighborhood interpolation weights. In *ICDM*, pages 43–52, 2007.

[19] Milena Mihail and Christos H. Papadimitriou. On the eigenvalue power law. In *RANDOM*, pages 254–262, 2002.

[20] Benjamin Recht, Maryam Fazel, and Pablo A. Parrilo. Guaranteed minimum-rank solutions of linear matrix equations via nuclear norm minimization, 2007.

